# Using the Equivalent Kernel to Understand Gaussian Process Regression

**Peter Sollich**
Dept of Mathematics
King's College London
Strand, London WC2R 2LS, UK
peter.sollich@kcl.ac.uk

**Christopher K. I. Williams**
School of Informatics
University of Edinburgh
5 Forrest Hill, Edinburgh EH1 2QL, UK
c.k.i.williams@ed.ac.uk

## Abstract

The equivalent kernel [1] is a way of understanding how Gaussian process regression works for large sample sizes based on a continuum limit. In this paper we show (1) how to approximate the equivalent kernel of the widely-used squared exponential (or Gaussian) kernel and related kernels, and (2) how analysis using the equivalent kernel helps to understand the learning curves for Gaussian processes.

Consider the supervised regression problem for a dataset $\mathcal{D}$ with entries $(\mathbf{x}_i, y_i)$ for $i = 1, \ldots, n$. Under Gaussian Process (GP) assumptions the predictive mean at a test point $\mathbf{x}_*$ is given by

$$\bar{f}(\mathbf{x}_*) = \mathbf{k}^\top(\mathbf{x}_*)(K + \sigma^2 I)^{-1}\mathbf{y}, \tag{1}$$

where $K$ denotes the $n \times n$ matrix of covariances between the training points with entries $k(\mathbf{x}_i, \mathbf{x}_j)$, $\mathbf{k}(\mathbf{x}_*)$ is the vector of covariances $k(\mathbf{x}_i, \mathbf{x}_*)$, $\sigma^2$ is the noise variance on the observations and $\mathbf{y}$ is a $n \times 1$ vector holding the training targets. See e.g. [2] for further details.

We can define a vector of functions $\mathbf{h}(\mathbf{x}_*) = (K + \sigma^2 I)^{-1}\mathbf{k}(\mathbf{x}_*)$. Thus we have $\bar{f}(\mathbf{x}_*) = \mathbf{h}^\top(\mathbf{x}_*)\mathbf{y}$, making it clear that the mean prediction at a point $\mathbf{x}_*$ is a linear combination of the target values $\mathbf{y}$. Gaussian process regression is thus a *linear smoother*, see [3, section 2.8] for further details. For a fixed test point $\mathbf{x}_*$, $\mathbf{h}(\mathbf{x}_*)$ gives the vector of weights applied to targets $\mathbf{y}$. Silverman [1] called $\mathbf{h}^\top(\mathbf{x}_*)$ the *weight function*.

Understanding the form of the weight function is made complicated by the matrix inversion of $K + \sigma^2 I$ and the fact that $K$ depends on the specific locations of the $n$ datapoints. Idealizing the situation one can consider the observations to be "smeared out" in $\mathbf{x}$-space at some constant density of observations. In this case analytic tools can be brought to bear on the problem, as shown below. By analogy to kernel smoothing Silverman [1] called the idealized weight function the *equivalent kernel* (EK).

The structure of the remainder of the paper is as follows: In section 1 we describe how to derive the equivalent kernel in Fourier space. Section 2 derives approximations for the EK for the squared exponential and other kernels. In section 3 we show how use the EK approach to estimate learning curves for GP regression, and compare GP regression to kernel regression using the EK.

# 1 Gaussian Process Regression and the Equivalent Kernel

It is well known (see e.g. [4]) that the posterior mean for GP regression can be obtained as the function which minimizes the functional

$$J[f] = \frac{1}{2}\|f\|_{\mathcal{H}}^2 + \frac{1}{2\sigma_n^2}\sum_{i=1}^{n}(y_i - f(\mathbf{x}_i))^2, \tag{2}$$

where $\|f\|_{\mathcal{H}}$ is the RKHS norm corresponding to kernel $k$. (However, note that the GP framework gives much more than just this mean prediction, for example the predictive variance and the marginal likelihood $p(\mathbf{y})$ of the data under the model.)

Let $\eta(\mathbf{x}) = \mathbb{E}[y|\mathbf{x}]$ be the target function for our regression problem and write $\mathbb{E}[(y - f(\mathbf{x}))^2] = \mathbb{E}[(y - \eta(\mathbf{x}))^2] + (\eta(\mathbf{x}) - f(\mathbf{x}))^2$. Using the fact that the first term on the RHS is independent of $f$ motivates considering a smoothed version of equation 2,

$$J_\rho[f] = \frac{\rho}{2\sigma^2}\int(\eta(\mathbf{x}) - f(\mathbf{x}))^2 d\mathbf{x} + \frac{1}{2}\|f\|_{\mathcal{H}}^2,$$

where $\rho$ has dimensions of the number of observations per unit of $\mathbf{x}$-space (length/area/volume etc. as appropriate). If we consider kernels that are stationary, $k(\mathbf{x}, \mathbf{x}') = k(\mathbf{x} - \mathbf{x}')$, the natural basis in which to analyse equation 1 is the Fourier basis of complex sinusoids so that $f(\mathbf{x})$ is represented as $\int \tilde{f}(\mathbf{s})e^{2\pi i\mathbf{s}\cdot\mathbf{x}}d\mathbf{s}$ and similarly for $\eta(\mathbf{x})$. Thus we obtain

$$J_\rho[f] = \frac{1}{2}\int\left(\frac{\rho}{\sigma^2}|\tilde{f}(\mathbf{s}) - \tilde{\eta}(\mathbf{s})|^2 + \frac{|\tilde{f}(\mathbf{s})|^2}{S(\mathbf{s})}\right)d\mathbf{s},$$

as $\|f\|_{\mathcal{H}}^2 = \int|\tilde{f}(\mathbf{s})|^2/S(\mathbf{s})d\mathbf{s}$ where $S(\mathbf{s})$ is the power spectrum of the kernel $k$, $S(\mathbf{s}) = \int k(\mathbf{x})e^{-2\pi i\mathbf{s}\cdot\mathbf{x}}d\mathbf{x}$. $J_\rho[f]$ can be minimized using calculus of variations to obtain $\tilde{f}(\mathbf{s}) = S(\mathbf{s})\eta(\mathbf{s})/(\sigma^2/\rho + S(\mathbf{s}))$ which is recognized as the convolution $f(\mathbf{x}_*) = \int h(\mathbf{x}_* - \mathbf{x})\eta(\mathbf{x})d\mathbf{x}$. Here the Fourier transform of the equivalent kernel $h(\mathbf{x})$ is

$$\tilde{h}(\mathbf{s}) = \frac{S(\mathbf{s})}{S(\mathbf{s}) + \sigma^2/\rho} = \frac{1}{1 + \sigma^2/(\rho S(\mathbf{s}))}. \tag{3}$$

The term $\sigma^2/\rho$ in the first expression for $\tilde{h}(\mathbf{s})$ corresponds to the power spectrum of a white noise process, whose delta-function covariance function becomes a constant in the Fourier domain. This analysis is known as Wiener filtering; see, e.g. [5, §14-1]. Notice that as $\rho \to \infty$, $h(\mathbf{x})$ tends to the delta function. If the input density is non-uniform the analysis above should be interpreted as computing the equivalent kernel for $np(\mathbf{x}) = \rho$. This approximation will be valid if the scale of variation of $p(\mathbf{x})$ is larger than the width of the equivalent kernel.

# 2 The EK for the Squared Exponential and Related Kernels

For certain kernels/covariance functions the EK $h(\mathbf{x})$ can be computed exactly by Fourier inversion. Examples include the Ornstein-Uhlenbeck process in $D = 1$ with covariance $k(x) = e^{-\alpha|x|}$ (see [5, p. 326]), splines in $D = 1$ corresponding to the regularizer $\|Pf\|^2 = \int(f^{(m)})^2 dx$ [1, 6], and the regularizer $\|Pf\|^2 = \int(\nabla^2 f)^2 d\mathbf{x}$ in two dimensions, where the EK is given in terms of the Kelvin function kei [7].

We now consider the commonly used squared exponential (SE) kernel $k(r) = \exp(-r^2/2\ell^2)$, where $r^2 = \|\mathbf{x} - \mathbf{x}'\|^2$. (This is sometimes called the Gaussian or radial basis function kernel.) Its Fourier transform is given by $S(\mathbf{s}) = (2\pi\ell^2)^{D/2}\exp(-2\pi^2\ell^2|\mathbf{s}|^2)$, where $D$ denotes the dimensionality of $\mathbf{x}$ (and $\mathbf{s}$) space.

From equation 3 we obtain

$$\tilde{h}_{\mathrm{SE}}(\mathbf{s}) = \frac{1}{1 + b\exp(2\pi^2\ell^2|\mathbf{s}|^2)},$$

where $b = \sigma^2/\rho(2\pi\ell^2)^{D/2}$. We are unaware of an exact result in this case, but the following initial approximation is simple but effective. For large $\rho$, $b$ will be small. Thus for small $s = |\mathbf{s}|$ we have that $\tilde{h}_{\mathrm{SE}} \simeq 1$, but for large $s$ it is approximately 0. The change takes place around the point $s_c$ where $b\exp(2\pi^2\ell^2 s_c^2) = 1$, i.e. $s_c^2 = \log(1/b)/2\pi^2\ell^2$. As $\exp(2\pi^2\ell^2 s^2)$ grows quickly with $s$, the transition of $\tilde{h}_{\mathrm{SE}}$ between 1 and 0 can be expected to be rapid, and thus be well-approximated by a step function.

**Proposition 1** *The approximate form of the equivalent kernel for the squared-exponential kernel in D-dimensions is given by*

$$h_{\mathrm{SE}}(r) = \left(\frac{s_c}{r}\right)^{D/2} J_{D/2}(2\pi s_c r).$$

**Proof**: $h_{\mathrm{SE}}(\mathbf{s})$ is a function of $s = |\mathbf{s}|$ only, and for $D > 1$ the Fourier integral can be simplified by changing to spherical polar coordinates and integrating out the angular variables to give

$$h_{\mathrm{SE}}(r) = 2\pi r \int_0^\infty \left(\frac{s}{r}\right)^{\nu+1} J_\nu(2\pi r s)\tilde{h}_{\mathrm{SE}}(s)\, ds \tag{4}$$

$$\simeq 2\pi r \int_0^{s_c} \left(\frac{s}{r}\right)^{\nu+1} J_\nu(2\pi r s)\, ds = \left(\frac{s_c}{r}\right)^{D/2} J_{D/2}(2\pi s_c r).$$

where $\nu = D/2 - 1$, $J_\nu(z)$ is a Bessel function of the first kind and we have used the identity $z^{\nu+1}J_\nu(z) = (d/dz)[z^{\nu+1}J_{\nu+1}(z)]$. $\square$

Note that in $D = 1$ by computing the Fourier transform of the boxcar function we obtain $h_{\mathrm{SE}}(x) = 2s_c\mathrm{sinc}(2\pi s_c x)$ where $\mathrm{sinc}(z) = \sin(z)/z$. This is consistent with Proposition 1 and $J_{1/2}(z) = (2/\pi z)^{1/2}\sin(z)$. The asymptotic form of the EK in $D = 2$ is shown in Figure 2(left) below.

Notice that $s_c$ scales as $(\log(\rho))^{1/2}$ so that the width of the EK (which is proportional to $1/s_c$) will decay very slowly as $\rho$ increases. In contrast for a spline of order $m$ (with power spectrum $\propto |\mathbf{s}|^{-2m}$) the width of the EK scales as $\rho^{-1/2m}$ [1].

If instead of $\mathbb{R}^D$ we consider the input set to be the unit circle, a stationary kernel can be periodized by the construction $k_p(x, x') = \sum_{n\in\mathbb{Z}} k(x - x' + 2n\pi)$. This kernel will be represented as a Fourier series (rather than with a Fourier transform) because of the periodicity. In this case the step function in Fourier space approximation would give rise to a Dirichlet kernel as the EK (see [8, section 4.4.3] for further details on the Dirichlet kernel).

We now show that the result of Proposition 1 is asymptotically exact for $\rho \to \infty$, and calculate the leading corrections for finite $\rho$. The scaling of the width of the EK as $1/s_c$ suggests writing $h_{\mathrm{SE}}(r) = (2\pi s_c)^D g(2\pi s_c r)$. Then from equation 4 and using the definition of $s_c$

$$g(z) = \frac{z}{s_c(2\pi s_c)^D} \int_0^\infty \left(\frac{2\pi s_c s}{z}\right)^{\nu+1} \frac{J_\nu(zs/s_c)}{1 + \exp[2\pi^2\ell^2(s^2 - s_c^2)]}\, ds$$

$$= z \int_0^\infty \left(\frac{u}{2\pi z}\right)^{\nu+1} \frac{J_\nu(zu)}{1 + \exp[2\pi^2\ell^2 s_c^2(u^2 - 1)]}\, du \tag{5}$$

where we have rescaled $s = s_c u$ in the second step. The value of $s_c$, and hence $\rho$, now enters only in the exponential via $a = 2\pi^2\ell^2 s_c^2$. For $a \to \infty$, the exponential tends to zero

for $u < 1$ and to infinity for $u > 1$. The factor $1/[1 + \exp(\ldots)]$ is therefore a step function $\Theta(1 - u)$ in the limit and Proposition 1 becomes exact, with $g_\infty(z) \equiv \lim_{a \to \infty} g(z) = (2\pi z)^{-D/2} J_{D/2}(z)$. To calculate corrections to this, one uses that for large but finite $a$ the difference $\Delta(u) = \{1 + \exp[a(u^2 - 1)]\}^{-1} - \Theta(1 - u)$ is non-negligible only in a range of order $1/a$ around $u = 1$. The other factors in the integrand of equation 5 can thus be Taylor-expanded around that point to give

$$g(z) = g_\infty(z) + z \sum_{k=0}^{\infty} \frac{I_k}{k!} \frac{d^k}{du^k} \left[ \left( \frac{u}{2\pi z} \right)^{\nu+1} J_\nu(zu) \right] \bigg|_{u=1}, \quad I_k = \int_0^\infty \Delta(u)(u - 1)^k \, du$$

The problem is thus reduced to calculating the integrals $I_k$. Setting $u = 1 + v/a$ one has

$$a^{k+1} I_k = \int_{-a}^0 \left[ \frac{1}{1 + \exp(v^2/a + 2v)} - 1 \right] v^k \, dv + \int_0^\infty \frac{v^k}{1 + \exp(v^2/a + 2v)} \, dv$$

$$= \int_0^a \frac{(-1)^{k+1} v^k}{1 + \exp(-v^2/a + 2v)} \, dv + \int_0^\infty \frac{v^k}{1 + \exp(v^2/a + 2v)} \, dv$$

In the first integral, extending the upper limit to $\infty$ gives an error that is exponentially small in $a$. Expanding the remaining $1/a$-dependence of the integrand one then gets, to leading order in $1/a$, $I_0 = c_0/a^2$, $I_1 = c_1/a^2$ while all $I_k$ with $k \geq 2$ are smaller by at least $1/a^2$. The numerical constants are $-c_0 = c_1 = \pi^2/24$. This gives, using that $(d/dz)[z^{\nu+1} J_\nu(z)] = z^\nu J_\nu(z) + z^{\nu+1} J_{\nu-1}(z) = (2\nu + 1)z^\nu J_\nu(z) - z^{\nu+1} J_{\nu+1}(z)$:

**Proposition 2** *The equivalent kernel for the squared-exponential kernel is given for large $\rho$ by $h_{\mathrm{SE}}(r) = (2\pi s_c)^D g(2\pi s_c r)$ with*

$$g(z) = \frac{1}{(2\pi z)^{\frac{D}{2}}} \left\{ J_{D/2}(z) + \frac{z}{a^2} \left[ (c_0 + c_1(D - 1)) J_{D/2-1}(z) - c_1 z J_{D/2}(z) \right] \right\} + O\left(\frac{1}{a^4}\right)$$

For e.g. $D = 1$ this becomes $g(z) = \pi^{-1} \{\sin(z)/z - \pi^2/(24a^2)[\cos(z) + z\sin(z)]\}$. Here and in general, by comparing the second part of the $1/a^2$ correction with the leading order term, one estimates that the correction is of relative size $z^2/a^2$. It will therefore provide a useful improvement as long as $z = 2\pi s_c r < a$; for larger $z$ the expansion in powers of $1/a$ becomes a poor approximation because the correction terms (of all orders in $1/a$) are comparable to the leading order.

## 2.1 Accuracy of the approximation

To evaluate the accuracy of the approximation we can compute the EK numerically as follows: Consider a dense grid of points in $\mathbb{R}^D$ with a sampling density $\rho_{\mathrm{grid}}$. For making predictions at the grid points we obtain the smoother matrix $K(K + \sigma_{\mathrm{grid}}^2 I)^{-1}$, where[1] $\sigma_{\mathrm{grid}}^2 = \sigma^2 \rho_{\mathrm{grid}}/\rho$, as per equation 1. Each row of this matrix is an approximation to the EK at the appropriate location, as this is the response to a $\mathbf{y}$ vector which is zero at all points except one. Note that in theory one should use a grid over the whole of $\mathbb{R}^D$ but in practice one can obtain an excellent approximation to the EK by only considering a grid around the point of interest as the EK typically decays with distance. Also, by only considering a finite grid one can understand how the EK is affected by edge effects.

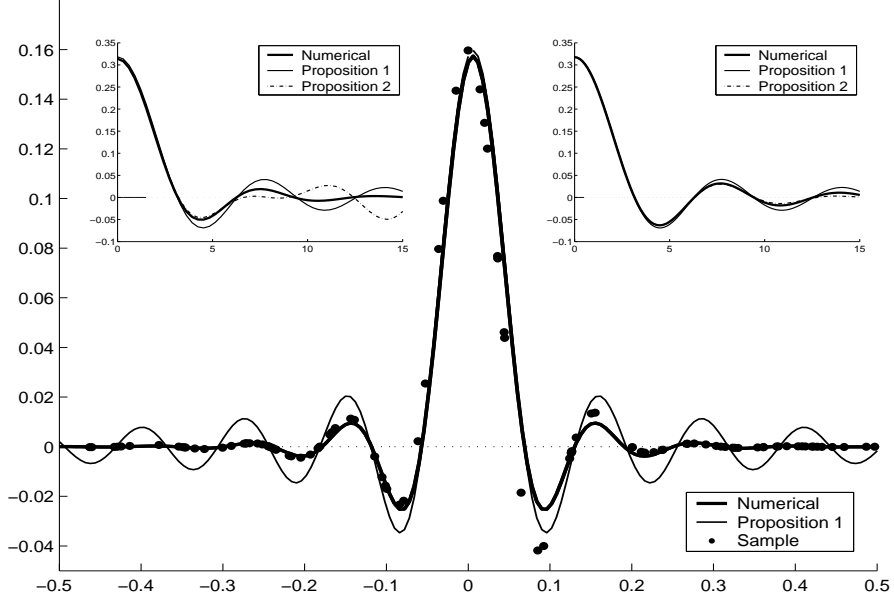

Figure 1: Main figure: plot of the weight function corresponding to $\rho = 100$ training points/unit length, plus the numerically computed equivalent kernel at $x = 0.0$ and the sinc approximation from Proposition 1. Insets: numerically evaluated $g(z)$ together with sinc and Proposition 2 approximations for $\rho = 100$ (left) and $\rho = 10^4$ (right).

Figure 1 shows plots of the weight function for $\rho = 100$, the EK computed on the grid as described above and the analytical sinc approximation. These are computed for parameter values of $\ell^2 = 0.004$ and $\sigma^2 = 0.1$, with $\rho_{\text{grid}}/\rho = 5/3$. To reduce edge effects, the interval $[-3/2, 3/2]$ was used for computations, although only the centre of this is shown in the figure. There is quite good agreement between the numerical computation and the analytical approximation, although the sidelobes decay more rapidly for the numerically computed EK. This is not surprising because the absence of a truly hard cutoff in Fourier space means one should expect less "ringing" than the analytical approximation predicts. The figure also shows good agreement between the weight function (based on the finite sample) and the numerically computed EK. The insets show the approximation of Proposition 2 to $g(z)$ for $\rho = 100$ ($a = 5.67$, left) and $\rho = 10^4$ ($a = 9.67$, right). As expected, the addition of the $1/a^2$-correction gives better agreement with the numerical result for $z < a$. Numerical experiments also show that the mean squared error between the numerically computed EK and the sinc approximation decreases like $1/\log(\rho)$. The is larger than the naïve estimate $(1/a^2)^2 \sim 1/(\log(\rho))^4$ based on the first correction term from Proposition 2, because the dominant part of the error comes from the region $z > a$ where the $1/a$ expansion breaks down.

## 2.2 Other kernels

Our analysis is not in fact restricted to the SE kernel. Consider an isotropic kernel, for which the power spectrum $S(\mathbf{s})$ depends on $s = |\mathbf{s}|$ only. Then we can again define from equation 3 an effective cutoff $s_c$ on the range of $s$ in the EK via $\sigma^2/\rho = S(s_c)$, so that $\tilde{h}(s) = [1 + S(s_c)/S(s)]^{-1}$. The EK will then have the limiting form given in Proposition 1 if $\tilde{h}(s)$ approaches a step function $\Theta(s_c - s)$, i.e. if it becomes infinitely "steep" around the point $s = s_c$ for $s_c \to \infty$. A quantitative criterion for this is that the slope

$|\tilde{h}'(s_c)|$ should become much larger than $1/s_c$, the inverse of the range of the step function. Since $\tilde{h}'(s) = S'(s)S(s_c)S^{-2}(s)[1 + S(s_c)/S(s)]^{-2}$, this is equivalent to requiring that $-s_c S'(s_c)/4S(s_c) \propto -d \log S(s_c)/d \log s_c$ must diverge for $s_c \to \infty$. The result of Proposition 1 therefore applies to *any* kernel whose power spectrum $S(s)$ decays more rapidly than any positive power of $1/s$.

A trivial example of a kernel obeying this condition would be a superposition of finitely many SE kernels with different lengthscales $\ell^2$; the asymptotic behaviour of $s_c$ is then governed by the smallest $\ell$. A less obvious case is the "rational quadratic" $k(r) = [1 + (r/l)^2]^{-(D+1)/2}$ which has an exponentially decaying power spectrum $S(s) \propto \exp(-2\pi\ell s)$. (This relationship is often used in the reverse direction, to obtain the power spectrum of the Ornstein-Uhlenbeck (OU) kernel $\exp(-r/\ell)$.) Proposition 1 then applies, with the width of the EK now scaling as $1/s_c \propto 1/\log(\rho)$.

The previous example is a special case of kernels which can be written as superpositions of SE kernels with a distribution $p(\ell)$ of lengthscales $\ell$, $k(r) = \int \exp(-r^2/2\ell^2)p(\ell)\,d\ell$. This is in fact the most general representation for an isotropic kernel which defines a valid covariance function in any dimension $D$, see [9, §2.10]. Such a kernel has power spectrum

$$S(s) = (2\pi)^{D/2} \int_0^\infty \ell^D \exp(-2\pi^2\ell^2 s^2)p(\ell)\,d\ell \tag{6}$$

and one easily verifies that the rational quadratic kernel, which has $S(s) \propto \exp(-2\pi\ell_0 s)$, is obtained for $p(\ell) \propto \ell^{-D-2} \exp(-\ell_0^2/2\ell^2)$. More generally, because the exponential factor in equation 6 acts like a cutoff for $\ell > 1/s$, one estimates $S(s) \sim \int_0^{1/s} \ell^D p(\ell)\,d\ell$ for large $s$. This will decay more strongly than any power of $1/s$ for $s \to \infty$ if $p(\ell)$ itself decreases more strongly than any power of $\ell$ for $\ell \to 0$. Any such choice of $p(\ell)$ will therefore yield a kernel to which Proposition 1 applies.

# 3  Understanding GP Learning Using the Equivalent Kernel

We now turn to using EK analysis to get a handle on average case learning curves for Gaussian processes. Here the setup is that a function $\eta$ is drawn from a Gaussian process, and we obtain $\rho$ noisy observations of $\eta$ per unit $\mathbf{x}$-space at random $\mathbf{x}$ locations. We are concerned with the mean squared error (MSE) between the GP prediction $\overline{f}$ and $\eta$. Averaging over the noise process, the $\mathbf{x}$-locations of the training data and the prior over $\eta$ we obtain the average MSE $\overline{\epsilon}$ as a function of $\rho$. See e.g. [10] and [11] for an overview of earlier work on GP learning curves.

To understand the asymptotic behaviour of $\overline{\epsilon}$ for large $\rho$, we now approximate the true GP predictions with the EK predictions from noisy data, given by $f_{\text{EK}}(\mathbf{x}) = \int h(\mathbf{x} - \mathbf{x}')y(\mathbf{x}')d\mathbf{x}'$ in the continuum limit of "smoothed out" input locations. We assume as before that $y = \text{target} + \text{noise}$, i.e. $y(\mathbf{x}) = \eta(\mathbf{x}) + \nu(\mathbf{x})$ where $\mathbb{E}[\nu(\mathbf{x})\nu(\mathbf{x}')] = (\sigma_*^2/\rho)\delta(\mathbf{x} - \mathbf{x}')$. Here $\sigma_*^2$ denotes the true noise variance, as opposed to the noise variance assumed in the EK; the scaling of $\sigma_*^2$ with $\rho$ is explained in footnote 1. For a fixed target $\eta$, the MSE is $\epsilon = (\int d\mathbf{x})^{-1} \int [\eta(\mathbf{x}) - f_{\text{EK}}(\mathbf{x})]^2 d\mathbf{x}$. Averaging over the noise process $\nu$ and target function $\eta$ gives in Fourier space

$$\overline{\epsilon} = \int \left\{ S_\eta(\mathbf{s})[1 - \tilde{h}(\mathbf{s})]^2 + (\sigma_*^2/\rho)\tilde{h}^2(\mathbf{s}) \right\} d\mathbf{s} = \frac{\sigma^2}{\rho} \int \frac{(\sigma^2/\rho)S_\eta(\mathbf{s})/S^2(\mathbf{s}) + \sigma_*^2/\sigma^2}{[1 + \sigma^2/(\rho S(\mathbf{s}))]^2}\,d\mathbf{s} \tag{7}$$

where $S_\eta(\mathbf{s})$ is the power spectrum of the prior over target functions. In the case $S(\mathbf{s}) = S_\eta(\mathbf{s})$ and $\sigma^2 = \sigma_*^2$ where the kernel is exactly matched to the structure of the target, equation 7 gives the Bayes error $\overline{\epsilon}_{\text{B}}$ and simplifies to $\overline{\epsilon}_{\text{B}} = (\sigma^2/\rho) \int [1 + \sigma^2/(\rho S(\mathbf{s}))]^{-1} d\mathbf{s}$ (see also [5, eq. 14-16]). Interestingly, this is just the analogue (for a continuous power spectrum of the kernel rather than a discrete set of eigenvalues) of the lower bound of [10]

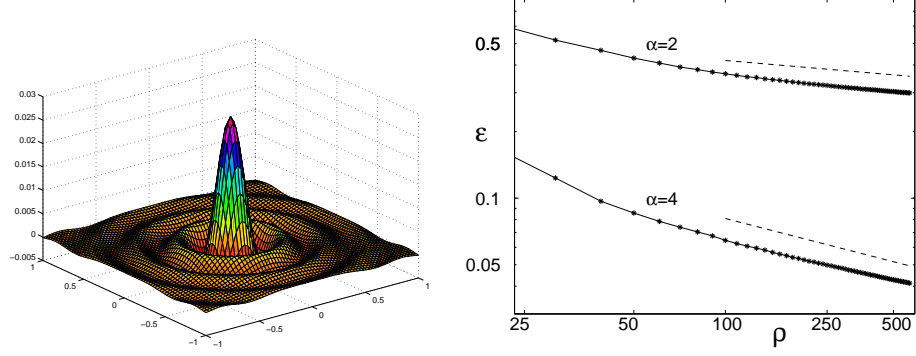

Figure 2: Left: plot of the asymptotic form of the EK $(s_c/r)J_1(2\pi s_c r)$ for $D = 2$ and $\rho = 1225$. Right: log-log plot of $\bar{\epsilon}$ against $\log(\rho)$ for the OU and Matern-class processes ($\alpha = 2, 4$ respectively). The dashed lines have gradients of $-1/2$ and $-3/2$ which are the predicted rates.

on the MSE of standard GP prediction from finite datasets. In experiments this bound provides a good approximation to the actual average MSE for large dataset size $n$ [11]. This supports our approach of using the EK to understand the learning behaviour of GP regression.

Treating the denominator in the expression for $\bar{\epsilon}_B$ again as a hard cutoff at $s = s_c$, which is justified for large $\rho$, one obtains for an SE target and learner $\bar{\epsilon} \approx \sigma^2 s_c/\rho \propto (\log(\rho))^{D/2}/\rho$. To get analogous predictions for the mismatched case, one can write equation 7 as

$$\bar{\epsilon} = \frac{\sigma_*^2}{\rho} \int \frac{[1 + \sigma^2/(\rho S(\mathbf{s}))] - \sigma^2/(\rho S(\mathbf{s}))}{[1 + \sigma^2/(\rho S(\mathbf{s}))]^2} \, d\mathbf{s} + \int \frac{S_\eta(\mathbf{s})}{[S(\mathbf{s})\rho/\sigma^2 + 1]^2} \, d\mathbf{s}.$$

The first integral is smaller than $(\sigma_*^2/\sigma^2)\bar{\epsilon}_B$ and can be neglected as long as $\bar{\epsilon} \gg \bar{\epsilon}_B$. In the second integral we can again make the cutoff approximation—though now with $s$ having to be *above* $s_c$ – to get the scaling $\bar{\epsilon} \propto \int_{s_c}^{\infty} s^{D-1} S_\eta(s) \, ds$. For target functions with a power-law decay $S_\eta(s) \propto s^{-\alpha}$ of the power spectrum at large $s$ this predicts $\bar{\epsilon} \propto s_c^{D-\alpha} \propto (\log(\rho))^{(D-\alpha)/2}$. So we generically get slow logarithmic learning, consistent with the observations in [12]. For $D = 1$ and an OU target ($\alpha = 2$) we obtain $\bar{\epsilon} \sim (\log(\rho))^{-1/2}$, and for the Matern-class covariance function $k(r) = (1 + r/\ell) \exp(-r/\ell)$ (which has power spectrum $\propto (3/\ell^2 + 4\pi^2 s^2)^{-2}$, so $\alpha = 4$) we get $\bar{\epsilon} \sim (\log(\rho))^{-3/2}$. These predictions were tested experimentally using a GP learner with SE covariance function ($\ell = 0.1$ and assumed noise level $\sigma^2 = 0.1$) against targets from the OU and Matern-class priors (with $\ell = 0.05$) and with noise level $\sigma_*^2 = 0.01$, averaging over 100 replications for each value of $\rho$. To demonstrate the predicted power-law dependence of $\bar{\epsilon}$ on $\log(\rho)$, in Figure 2(right) we make a log-log plot of $\bar{\epsilon}$ against $\log(\rho)$. The dashed lines show the gradients of $-1/2$ and $-3/2$ and we observe good agreement between experimental and theoretical results for large $\rho$.

## 3.1 Using the Equivalent Kernel in Kernel Regression

Above we have used the EK to understand how standard GP regression works. One could alternatively envisage using the EK to perform kernel regression, on given finite data sets, producing a prediction $\rho^{-1} \sum_i h(\mathbf{x}_* - \mathbf{x}_i)y_i$ at $\mathbf{x}_*$. Intuitively this seems appealing as a cheap alternative to full GP regression, particularly for kernels such as the SE where the EK can be calculated analytically, at least to a good approximation. We now analyze briefly how such an EK predictor would perform compared to standard GP prediction.

Letting $\langle \cdot \rangle$ denote averaging over noise, training input points and the test point and setting $f_\eta(\mathbf{x}_*) = \int h(\mathbf{x}, \mathbf{x}_*)\eta(\mathbf{x})d\mathbf{x}$, the average MSE of the EK predictor is

$$\bar{\epsilon}_{\text{pred}} = \langle [\eta(\mathbf{x}) - (1/\rho)\sum_i h(\mathbf{x}, \mathbf{x}_i)y_i]^2 \rangle$$

$$= \langle [\eta(\mathbf{x}) - f_\eta(\mathbf{x})]^2 + \frac{\sigma_*^2}{\rho} \int h^2(\mathbf{x}, \mathbf{x}')d\mathbf{x}' + \frac{1}{\rho}\langle \int h^2(\mathbf{x}, \mathbf{x}')\eta^2(\mathbf{x}')d\mathbf{x}' \rangle - \frac{1}{\rho}\langle f_\eta^2(\mathbf{x}) \rangle$$

$$= \frac{\sigma^2}{\rho} \int \frac{(\sigma^2/\rho)S_\eta(\mathbf{s})/S^2(\mathbf{s}) + \sigma_*^2/\sigma^2}{[1 + \sigma^2/(\rho S(\mathbf{s}))]^2} d\mathbf{s} + \frac{\langle \eta^2 \rangle}{\rho} \int \frac{d\mathbf{s}}{[1 + \sigma^2/(\rho S(\mathbf{s}))]^2}$$

Here we have set $\langle \eta^2 \rangle = (\int d\mathbf{x})^{-1} \int \eta^2(\mathbf{x}) d\mathbf{x} = \int S_\eta(\mathbf{s}) d\mathbf{s}$ for the spatial average of the squared target amplitude. Taking the matched case, $(S_\eta(\mathbf{s}) = S(\mathbf{s})$ and $\sigma_*^2 = \sigma^2)$ as an example, the first term (which is the one we get for the prediction from "smoothed out" training inputs, see eq. 7) is of order $\sigma^2 s_c^D/\rho$, while the second one is $\sim \langle \eta^2 \rangle s_c^D/\rho$. Thus both terms scale in the same way, but the ratio of the second term to the first is the signal-to-noise ratio $\langle \eta^2 \rangle/\sigma^2$, which in practice is often large. The EK predictor will then perform significantly worse than standard GP prediction, by a roughly constant factor, and we have confirmed this prediction numerically. This result is somewhat surprising given the good agreement between the weight function $\mathbf{h}(\mathbf{x}_*)$ and the EK that we saw in figure 1, leading to the conclusion that the detailed structure of the weight function is important for optimal prediction from finite data sets.

In summary, we have derived accurate approximations for the equivalent kernel (EK) of GP regression with the widely used squared exponential kernel, and have shown that the same analysis in fact extends to a whole class of kernels. We have also demonstrated that EKs provide a simple means of understanding the learning behaviour of GP regression, even in cases where the learner's covariance function is not well matched to the structure of the target function. In future work, it will be interesting to explore in more detail the use of the EK in kernel smoothing. This is suboptimal compared to standard GP regression as we saw. However, it does remain feasible even for very large datasets, and may then be competitive with sparse methods for approximating GP regression. From the theoretical point of view, the average error of the EK predictor which we calculated may also provide the basis for useful upper bounds on GP learning curves.

Acknowledgments: This work was supported in part by the IST Programme of the European Community, under the PASCAL Network of Excellence, IST-2002-506778. This publication only reflects the authors' views.

## Footnotes

[1]To understand this scaling of $\sigma_{\mathrm{grid}}^2$ consider the case where $\rho_{\mathrm{grid}} > \rho$ which means that the effective variance at each of the $\rho_{\mathrm{grid}}$ points per unit $\mathbf{x}$-space is larger, but as there are correspondingly more points this effect cancels out. This can be understood by imagining the situation where there are $\rho_{\mathrm{grid}}/\rho$ independent Gaussian observations with variance $\sigma_{\mathrm{grid}}^2$ at a single $\mathbf{x}$-point; this would be equivalent to one Gaussian observation with variance $\sigma^2$. In effect the $\rho$ observations per unit $\mathbf{x}$-space have been smoothed out uniformly.

## References

[1] B. W. Silverman. *Annals of Statistics*, 12:898–916, 1984.

[2] C. K. I. Williams. In M. I. Jordan, editor, *Learning in Graphical Models*, pages 599–621. Kluwer Academic, 1998.

[3] T. J. Hastie and R. J. Tibshirani. *Generalized Additive Models*. Chapman and Hall, 1990.

[4] F. Girosi, M. Jones, and T. Poggio. *Neural Computation*, 7(2):219–269, 1995.

[5] A. Papoulis. *Probability, Random Variables, and Stochastic Processes*. McGraw-Hill, New York, 1991. Third Edition.

[6] C. Thomas-Agnan. *Numerical Algorithms*, 13:21–32, 1996.

[7] T. Poggio, H. Voorhees, and A. Yuille. Tech. Report AI Memo 833, MIT AI Laboratory, 1985.

[8] B. Schölkopf and A. Smola. *Learning with Kernels*. MIT Press, 2002.

[9] M. L. Stein. *Interpolation of Spatial Data*. Springer-Verlag, New York, 1999.

[10] M. Opper and F. Vivarelli. In *NIPS 11*, pages 302–308, 1999.

[11] P. Sollich and A. Halees. *Neural Computation*, 14:1393–1428, 2002.

[12] P. Sollich. In *NIPS 14*, pages 519–526, 2002.
